# Feature Selection for SVMs

**J. Weston[†], S. Mukherjee[††], O. Chapelle[*], M. Pontil[††]**
**T. Poggio[††], V. Vapnik[*,†††]**
[†] Barnhill BioInformatics.com, Savannah, Georgia, USA.
[††] CBCL MIT, Cambridge, Massachusetts, USA.
[*] AT&T Research Laboratories, Red Bank, USA.
[†††] Royal Holloway, University of London, Egham, Surrey, UK.

## Abstract

We introduce a method of feature selection for Support Vector Machines. The method is based upon finding those features which minimize bounds on the leave-one-out error. This search can be efficiently performed via gradient descent. The resulting algorithms are shown to be superior to some standard feature selection algorithms on both toy data and real-life problems of face recognition, pedestrian detection and analyzing DNA microarray data.

## 1   Introduction

In many supervised learning problems feature selection is important for a variety of reasons: generalization performance, running time requirements, and constraints and interpretational issues imposed by the problem itself.

In classification problems we are given $\ell$ data points $\mathbf{x}_i \in \mathbb{R}^n$ labeled $y \in \pm 1$ drawn i.i.d from a probability distribution $P(\mathbf{x}, y)$. We would like to select a subset of features while preserving or improving the discriminative ability of a classifier. As a brute force search of all possible features is a combinatorial problem one needs to take into account both the quality of solution and the computational expense of any given algorithm.

Support vector machines (SVMs) have been extensively used as a classification tool with a great deal of success from object recognition [5, 11] to classification of cancer morphologies [10] and a variety of other areas, see e.g [13] . In this article we introduce feature selection algorithms for SVMs. The methods are based on minimizing generalization bounds via gradient descent and are feasible to compute. This allows several new possibilities: one can speed up time critical applications (e.g object recognition) and one can perform feature discovery (e.g cancer diagnosis). We also show how SVMs can perform badly in the situation of many irrelevant features, a problem which is remedied by using our feature selection approach.

The article is organized as follows. In section 2 we describe the feature selection problem, in section 3 we review SVMs and some of their generalization bounds and in section 4 we introduce the new SVM feature selection method. Section 5 then describes results on toy and real life data indicating the usefulness of our approach.

## 2    The Feature Selection problem

The feature selection problem can be addressed in the following two ways: (1) given a fixed $m \ll n$, find the $m$ features that give the smallest expected generalization error; or (2) given a maximum allowable generalization error $\gamma$, find the smallest $m$. In both of these problems the expected generalization error is of course unknown, and thus must be estimated. In this article we will consider problem (1). Note that choices of $m$ in problem (1) can usually can be reparameterized as choices of $\gamma$ in problem (2).

Problem (1) is formulated as follows. Given a fixed set of functions $y = f(\mathbf{x}, \alpha)$ we wish to find a preprocessing of the data $\mathbf{x} \mapsto (\mathbf{x} * \sigma)$, $\sigma \in \{0, 1\}^n$, and the parameters $\alpha$ of the function $f$ that give the minimum value of

$$\tau(\sigma, \alpha) = \int V(y, f((\mathbf{x} * \sigma), \alpha)) dP(\mathbf{x}, y) \tag{1}$$

subject to $\|\sigma\|_0 = m$, where $P(\mathbf{x}, y)$ is unknown, $x * \sigma = (x_1\sigma_1, \ldots, x_n\sigma_n)$ denotes an elementwise product, $V(\cdot, \cdot)$ is a loss functional and $\| \cdot \|_0$ is the 0-norm.

In the literature one distinguishes between two types of method to solve this problem: the so-called filter and wrapper methods [2]. Filter methods are defined as a preprocessing step to induction that can remove irrelevant attributes before induction occurs, and thus wish to be valid for any set of functions $f(\mathbf{x}, \alpha)$. For example one popular filter method is to use Pearson correlation coefficients.

The wrapper method, on the other hand, is defined as a search through the space of feature subsets using the estimated accuracy from an induction algorithm as a measure of goodness of a particular feature subset. Thus, one approximates $\tau(\sigma, \alpha)$ by minimizing

$$\tau_{wrap}(\sigma, \alpha) = \min_{\sigma} \tau_{alg}(\sigma) \tag{2}$$

subject to $\sigma \in \{0, 1\}^n$ where $\tau_{alg}$ is a learning algorithm trained on data preprocessed with fixed $\sigma$. Wrapper methods can provide more accurate solutions than filter methods [9], but in general are more computationally expensive since the induction algorithm $\tau_{alg}$ must be evaluated over each feature set (vector $\sigma$) considered, typically using performance on a hold out set as a measure of goodness of fit.

In this article we introduce a feature selection algorithm for SVMs that takes advantage of the performance increase of wrapper methods whilst avoiding their computational complexity. Note, some previous work on feature selection for SVMs does exist, however results have been limited to linear kernels [3, 7] or linear probabilistic models [8]. Our approach can be applied to nonlinear problems. In order to describe this algorithm, we first review the SVM method and some of its properties.

## 3    Support Vector Learning

Support Vector Machines [13] realize the following idea: they map $\mathbf{x} \in \mathbb{R}^n$ into a high (possibly infinite) dimensional space and construct an optimal hyperplane in this space. Different mappings $\mathbf{x} \mapsto \Phi(\mathbf{x}) \in \mathcal{H}$ construct different SVMs.

The mapping $\Phi(\cdot)$ is performed by a kernel function $K(\cdot, \cdot)$ which defines an inner product in $\mathcal{H}$. The decision function given by an SVM is thus:

$$f(\mathbf{x}) = w \cdot \Phi(\mathbf{x}) + b = \sum_i \alpha_i^0 y_i K(\mathbf{x}_i, \mathbf{x}) + b. \tag{3}$$

The optimal hyperplane is the one with the maximal distance (in $\mathcal{H}$ space) to the closest image $\Phi(\mathbf{x}_i)$ from the training data (called the maximal margin). This reduces to maximizing

the following optimization problem:

$$W^2(\alpha) = \sum_{i=1}^{\ell} \alpha_i - \frac{1}{2} \sum_{i,j=1}^{\ell} \alpha_i \alpha_j y_i y_j K(\mathbf{x}_i, \mathbf{x}_j) \tag{4}$$

under constraints $\sum_{i=1}^{\ell} \alpha_i y_i = 0$ and $\alpha_i \geq 0$, $i = 1, ..., \ell$. For the non-separable case one can quadratically penalize errors with the modified kernel $K \leftarrow K + \frac{1}{\lambda} I$ where $I$ is the identity matrix and $\lambda$ a constant penalizing the training errors (see [4] for reasons for this choice).

Suppose that the size of the maximal margin is $M$ and the images $\Phi(\mathbf{x}_1), ..., \Phi(\mathbf{x}_\ell)$ of the training vectors are within a sphere of radius $R$. Then the following holds true [13].

**Theorem 1** *If images of training data of size $\ell$ belonging to a sphere of size $R$ are separable with the corresponding margin $M$, then the expectation of the error probability has the bound*

$$EP_{err} \leq \frac{1}{\ell} E \left\{ \frac{R^2}{M^2} \right\} = \frac{1}{\ell} E \left\{ R^2 W^2(\alpha^0) \right\}, \tag{5}$$

*where expectation is taken over sets of training data of size $\ell$.*

This theorem justifies the idea that the performance depends on the ratio $E\{R^2/M^2\}$ and not simply on the large margin $M$, where $R$ is controlled by the mapping function $\Phi(\cdot)$.

Other bounds also exist, in particular Vapnik and Chapelle [4] derived an estimate using the concept of the *span* of support vectors.

**Theorem 2** *Under the assumption that the set of support vectors does not change when removing the example p*

$$EP_{err}^{\ell-1} \leq \frac{1}{\ell} E \sum_{p=1}^{\ell} \Psi \left( \frac{\alpha_p^0}{(K_{SV}^{-1})_{pp}} - 1 \right) \tag{6}$$

*where $\Psi$ is the step function, $K_{SV}$ is the matrix of dot products between support vectors, $p_{err}^{\ell-1}$ is the probability of test error for the machine trained on a sample of size $\ell - 1$ and the expectations are taken over the random choice of the sample.*

## 4 Feature Selection for SVMs

In the problem of feature selection we wish to minimize equation (1) over $\sigma$ and $\alpha$. The support vector method attempts to find the function from the set $f(\mathbf{x}, w, b) = w \cdot \Phi(\mathbf{x}) + b$ that minimizes generalization error. We first enlarge the set of functions considered by the algorithm to $f(\mathbf{x}, w, b, \sigma) = w \cdot \Phi(\mathbf{x} * \sigma) + b$. Note that the mapping $\Phi_\sigma(\mathbf{x}) = \Phi(\mathbf{x} * \sigma)$ can be represented by choosing the kernel function $K_\sigma$ in equations (3) and (4):

$$K_\sigma(\mathbf{x}, \mathbf{y}) = K((\mathbf{x} * \sigma), (\mathbf{y} * \sigma)) = (\Phi_\sigma(\mathbf{x}) \cdot \Phi_\sigma(\mathbf{y})) \tag{7}$$

for any $K$. Thus for these kernels the bounds in Theorems (1) and (2) still hold. Hence, to minimize $\tau(\sigma, \alpha)$ over $\alpha$ and $\sigma$ we minimize the wrapper functional $\tau_{wrap}$ in equation (2) where $\tau_{alg}$ is given by the equations (5) or (6) choosing a fixed value of $\sigma$ implemented by the kernel (7). Using equation (5) one minimizes over $\sigma$:

$$R^2 W^2(\sigma) = R^2(\sigma) W^2(\alpha^0, \sigma) \tag{8}$$

where the radius $R$ for kernel $K_\sigma$ can be computed by maximizing (see, e.g [13]):

$$R^2(\sigma) = \max_\beta \sum_i \beta_i K_\sigma(\mathbf{x}_i, \mathbf{x}_i) - \sum_{i,j} \beta_i \beta_j K_\sigma(\mathbf{x}_i, \mathbf{x}_j) \tag{9}$$

subject to $\sum_i \beta_i = 1$, $\beta_i \geq 0$, $i = 1, \ldots, \ell$, and $W^2(\alpha^0, \sigma)$ is defined by the maximum of functional (4) using kernel (7). In a similar way, one can minimize the *span* bound over $\sigma$ instead of equation (8).

Finding the minimum of $R^2 W^2$ over $\sigma$ requires searching over all possible subsets of $n$ features which is a combinatorial problem. To avoid this problem classical methods of search include greedily adding or removing features (forward or backward selection) and hill climbing. All of these methods are expensive to compute if $n$ is large.

As an alternative to these approaches we suggest the following method: approximate the binary valued vector $\sigma \in \{0,1\}^n$, with a real valued vector $\sigma \in \mathbb{R}^n$. Then, to find the optimum value of $\sigma$ one can minimize $R^2 W^2$, or some other differentiable criterion, by gradient descent. As explained in [4] the derivative of our criterion is:

$$\frac{\partial R^2 W^2(\sigma)}{\partial \sigma_k} = R^2(\sigma) \frac{\partial W^2(\alpha^0, \sigma)}{\partial \sigma_k} + W^2(\alpha^0, \sigma) \frac{\partial R^2(\sigma)}{\partial \sigma_k} \tag{10}$$

$$\frac{\partial R^2(\sigma)}{\partial \sigma_k} = \sum_i \beta_i^0 \frac{\partial K_\sigma(\mathbf{x}_i, \mathbf{x}_j)}{\partial \sigma_k} - \sum_{i,j} \beta_i^0 \beta_j^0 y_i y_j \frac{\partial K_\sigma(\mathbf{x}_i, \mathbf{x}_j)}{\partial \sigma_k} \tag{11}$$

$$\frac{\partial W^2(\alpha^0, \sigma)}{\partial \sigma_k} = -\sum_{i,j} \alpha_i^0 \alpha_j^0 y_i y_j \frac{\partial K_\sigma(\mathbf{x}_i, \mathbf{x}_j)}{\partial \sigma_k}. \tag{12}$$

We estimate the minimum of $\tau(\sigma, \alpha)$ by minimizing equation (8) in the space $\sigma \in \mathbb{R}^n$ using the gradients (10) with the following extra constraint which approximates integer programming:

$$R^2 W^2(\sigma) + \lambda \sum_i (\sigma_i)^p \tag{13}$$

subject to $\sum_i \sigma_i = m$, $\sigma_i \geq 0$, $i = 1, \ldots, \ell$.

For large enough $\lambda$ as $p \to 0$ only $m$ elements of $\sigma$ will be nonzero, approximating optimization problem $\tau(\sigma, \alpha)$. One can further simplify computations by considering a stepwise approximation procedure to find $m$ features. To do this one can minimize $R^2 W^2(\sigma)$ with $\sigma$ unconstrained. One then sets the $q \ll n$ smallest values of $\sigma$ to zero, and repeats the minimization until only $m$ nonzero elements of $\sigma$ remain. This can mean repeatedly training a SVM just a few times, which can be fast.

## 5 Experiments

### 5.1 Toy data

We compared standard SVMs, our feature selection algorithms and three classical filter methods to select features followed by SVM training. The three filter methods chose the $m$ largest features according to: Pearson correlation coefficients, the Fisher criterion score[1], and the Kolmogorov-Smirnov test[2]). The Pearson coefficients and Fisher criterion cannot model nonlinear dependencies.

In the two following artificial datasets our objective was to assess the ability of the algorithm to select a small number of target features in the presence of irrelevant and redundant features.

**Linear problem** Six dimensions of 202 were relevant. The probability of $y = 1$ or $-1$ was equal. The first three features $\{x_1, x_2, x_3\}$ were drawn as $x_i = yN(i, 1)$ and the second three features $\{x_4, x_5, x_6\}$ were drawn as $x_i = N(0, 1)$ with a probability of 0.7, otherwise the first three were drawn as $x_i = N(0, 1)$ and the second three as $x_i = yN(i - 3, 1)$. The remaining features are noise $x_i = N(0, 20)$, $i = 7, \ldots, 202$.

**Nonlinear problem** Two dimensions of 52 were relevant. The probability of $y = 1$ or $-1$ was equal. The data are drawn from the following: if $y = -1$ then $\{x_1, x_2\}$ are drawn from $N(\mu_1, \Sigma)$ or $N(\mu_2, \Sigma)$ with equal probability, $\mu_1 = \{-\frac{3}{4}, -3\}$ and $\mu_2 = \{\frac{3}{4}, 3\}$ and $\Sigma = I$, if $y = 1$ then $\{x_1, x_2\}$ are drawn again from two normal distributions with equal probability, with $\mu_1 = \{3, -3\}$ and $\mu_2 = \{-3, 3\}$ and the same $\Sigma$ as before. The rest of the features are noise $x_i = N(0, 20)$, $i = 3, \ldots, 52$.

In the linear problem the first six features have redundancy and the rest of the features are irrelevant. In the nonlinear problem all but the first two features are irrelevant.

We used a linear SVM for the linear problem and a second order polynomial kernel for the nonlinear problem. For the filter methods and the SVM with feature selection we selected the 2 best features.

The results are shown in Figure (1) for various training set sizes, taking the average test error on 500 samples over 30 runs of each training set size. The Fisher score (not shown in graphs due to space constraints) performed almost identically to correlation coefficients.

In both problems standard SVMs perform poorly: in the linear example using $\ell = 500$ points one obtains a test error of 13% for SVMs, which should be compared to a test error of 3% with $\ell = 50$ using our methods. Our SVM feature selection methods also outperformed the filter methods, with forward selection being marginally better than gradient descent. In the nonlinear problem, among the filter methods only the Kolmogorov-Smirnov test improved performance over standard SVMs.

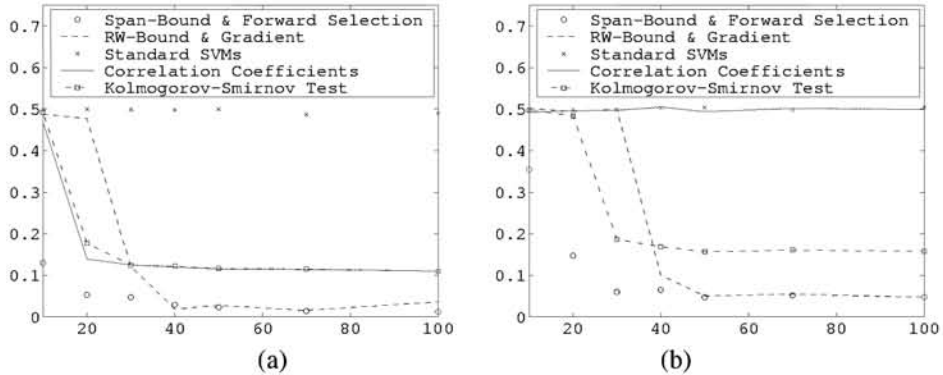

Figure 1: A comparison of feature selection methods on (a) a linear problem and (b) a nonlinear problem both with many irrelevant features. The $x$-axis is the number of training points, and the $y$-axis the test error as a fraction of test points.

## 5.2  Real-life data

For the following problems we compared minimizing $R^2W^2$ via gradient descent to the Fisher criterion score.

**Face detection** The face detection experiments described in this section are for the system introduced in [12, 5]. The training set consisted of $2, 429$ positive images of frontal faces of

size 19x19 and $13,229$ negative images not containing faces. The test set consisted of 105 positive images and $2,000,000$ negative images. A wavelet representation of these images [5] was used, which resulted in $1,740$ coefficients for each image.

Performance of the system using all coefficients, 725 coefficients, and 120 coefficients is shown in the ROC curve in figure (2a). The best results were achieved using all features, however $R^2W^2$ outperfomed the Fisher score. In this case feature selection was not useful for eliminating irrelevant features, but one could obtain a solution with comparable performance but reduced complexity, which could be important for time critical applications.

**Pedestrian detection** The pedestrian detection experiments described in this section are for the system introduced in [11]. The training set consisted of 924 positive images of people of size 128x64 and $10,044$ negative images not containing pedestrians. The test set consisted of 124 positive images and $800,000$ negative images. A wavelet representation of these images [5, 11] was used, which resulted in $1,326$ coefficients for each image.

Performance of the system using all coefficients and 120 coefficients is shown in the ROC curve in figure (2b). The results showed the same trends that were observed in the face recognition problem.

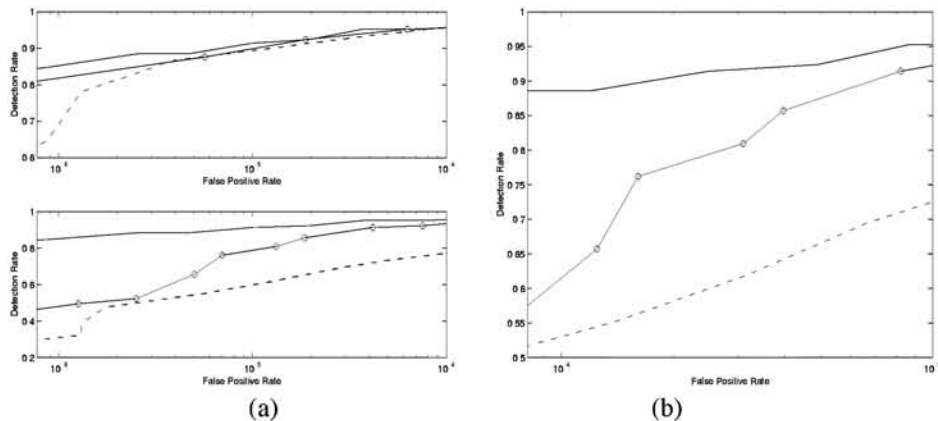

(a)     (b)

Figure 2: The solid line is using all features, the solid line with a circle is our feature selection method (minimizing $R^2W^2$ by gradient descent) and the dotted line is the Fisher score. (a)The top ROC curves are for 725 features and the bottom one
for 120 features for face detection. (b) ROC curves using all features and 120 features for pedestrian detection.

**Cancer morphology classification** For DNA microarray data analysis one needs to determine the relevant genes in discrimination as well as discriminate accurately. We look at two leukemia discrimination problems [6, 10] and a colon cancer problem [1] (see also [7] for a treatment of both of these problems).

The first problem was classifying myeloid and lymphoblastic leukemias based on the expression of 7129 genes. The training set consists of 38 examples and the test set of 34 examples. Using all genes a linear SVM makes 1 error on the test set. Using 20 genes 0 errors are made for $R^2W^2$ and 3 errors are made using the Fisher score. Using 5 genes 1 error is made for $R^2W^2$ and 5 errors are made for the Fisher score. The method of [6] performs comparably to the Fisher score.

The second problem was discriminating B versus T cells for lymphoblastic cells [6]. Standard linear SVMs make 1 error for this problem. Using 5 genes 0 errors are made for $R^2W^2$ and 3 errors are made using the Fisher score.

In the colon cancer problem [1] 62 tissue samples probed by oligonucleotide arrays contain 22 normal and 40 colon cancer tissues that must be discriminated based upon the expression of 2000 genes. Splitting the data into a training set of 50 and a test set of 12 in 50 separate trials we obtained a test error of 13% for standard linear SVMs. Taking 15 genes for each feature selection method we obtained 12.8% for $R^2W^2$, 17.0% for Pearson correlation coefficients, 19.3% for the Fisher score and 19.2% for the Kolmogorov-Smirnov test. Our method is only worse than the best filter method in 8 of the 50 trials.

## 6 Conclusion

In this article we have introduced a method to perform feature selection for SVMs. This method is computationally feasible for high dimensional datasets compared to existing wrapper methods, and experiments on a variety of toy and real datasets show superior performance to the filter methods tried. This method, amongst other applications, speeds up SVMs for time critical applications (e.g pedestrian detection), and makes possible feature discovery (e.g gene discovery). Secondly, in simple experiments we showed that SVMs can indeed suffer in high dimensional spaces where many features are irrelevant. Our method provides one way to circumvent this naturally occuring, complex problem.

## Footnotes

[1] $F(r) = \left| \frac{\mu_r^+ - \mu_r^-}{\sigma_r^{+2} + \sigma_r^{-2}} \right|$, where $\mu_r^\pm$ is the mean value for the $r$-th feature in the positive and negative classes and $\sigma_r^{\pm^2}$ is the standard deviation

[2] $\text{KS}_{tst}(r) = \sqrt{\ell} \sup \left( \hat{P}\{X \leq f_r\} - \hat{P}\{X \leq f_r, y_r = 1\} \right)$ where $f_r$ denotes the $r$-th feature from each training example, and $\hat{P}$ is the corresponding empirical distribution.

## References

[1] U. Alon, N. Barkai, D. Notterman, K. Gish, S. Ybarra, D. Mack, and A. Levine. Broad patterns of gene expression revealed by clustering analysis of tumor and normal colon cancer tissues probed by oligonucleotide arrays. *Cell Biology*, 96:6745–6750, 1999.

[2] A. Blum and P. Langley. Selection of relevant features and examples in machine learning. *Artificial Intelligence, 97:245–271,*, 1997.

[3] P. S. Bradley and O. L. Mangasarian. Feature selection via concave minimization and support vector machines. In *Proc. 13th International Conference on Machine Learning*, pages 82–90, San Francisco, CA, 1998.

[4] O. Chapelle, V. Vapnik, O. Bousquet, and S. Mukherjee. Choosing kernel parameters for support vector machines. *Machine Learning*, 2000.

[5] T. Evgeniou, M. Pontil, C. Papageorgiou, and T. Poggio. Image representations for object detection using kernel classifiers. In *Asian Conference on Computer Vision*, 2000.

[6] T. Golub, D. Slonim, P. Tamayo, C. Huard, M. Gaasenbeek, J. Mesirov, H. Coller, M. Loh, J. Downing, M. Caligiuri, C. D. Bloomfield, and E. S. Lander. Molecular classification of cancer: Class discovery and class prediction by gene expression monitoring. *Science*, 286:531–537, 1999.

[7] I. Guyon, J. Weston, S. Barnhill, and V. Vapnik. Gene selection for cancer classification using support vector machines. *Machine Learning*, 2000.

[8] T. Jebara and T. Jaakkola. Feature selection and dualities in maximum entropy discrimination. In *Uncertainity In Artificial Intellegence*, 2000.

[9] J. Kohavi. Wrappers for feature subset selection. *AIJ issue on relevance*, 1995.

[10] S. Mukherjee, P. Tamayo, D. Slonim, A. Verri, T. Golub, J. Mesirov, and T. Poggio. Support vector machine classification of microarray data. AI Memo 1677, Massachusetts Institute of Technology, 1999.

[11] M. Oren, C. Papageorgiou, P. Sinha, E. Osuna, and T. Poggio. Pedestrian detection using wavelet templates. In *Proc. Computer Vision and Pattern Recognition*, pages 193–199, Puerto Rico, June 16–20 1997.

[12] C. Papageorgiou, M. Oren, and T. Poggio. A general framework for object detection. In *International Conference on Computer Vision*, Bombay, India, January 1998.

[13] V. Vapnik. *Statistical Learning Theory*. John Wiley and Sons, New York, 1998.
